# Neural Network Definitions of Highly Predictable Protein Secondary Structure Classes

**Alan Lapedes**
Complex Systems Group (T13)
LANL, MS B213 Los Alamos N.M. 87545
and The Santa Fe Institute, Santa Fe, New Mexico
Evan Steeg
Department of Computer Science
University of Toronto, Toronto, Canada
Robert Farber
Complex Systems Group (T13)
LANL, MS B213 Los Alamos N.M. 87545

## Abstract

We use two co-evolving neural networks to determine *new* classes of protein secondary structure which are significantly more predictable from local amino sequence than the conventional secondary structure classification. Accurate prediction of the conventional secondary structure classes: alpha helix, beta strand, and coil, from primary sequence has long been an important problem in computational molecular biology. Neural networks have been a popular method to attempt to predict these conventional secondary structure classes. Accuracy has been disappointingly low. The algorithm presented here uses neural networks to similtaneously examine both sequence and structure data, and to evolve new classes of secondary structure that can be predicted from sequence with significantly higher accuracy than the conventional classes. These new classes have both similarities to, and differences with the conventional alpha helix, beta strand and coil.

The conventional classes of protein secondary structure, alpha helix and beta sheet, were first introduced in 1951 by Linus Pauling and Robert Corey [Pauling, 1951] on the basis of molecular modeling. Prediction of secondary structure from the amino acid sequence has long been an important problem in computational molecular biology. There have been numerous attempts to predict locally defined secondary structure classes using only a local window of sequence information. The prediction methodology ranges from a combination of statistical and rule-based methods [Chou, 1978] to neural net methods [Qian, 1988], [Maclin, 1992], [Kneller, 1990], [Stolorz, 1992]. Despite a variety of intense efforts, the accuracy of prediction of conventional secondary structure is still distressingly low.

In this paper we will use neural networks to generalize the notion of protein secondary structure and to find new classes of structure that are significantly more predictable. We define protein "secondary structure" to be any classification of protein structure that can be defined using only local "windows" of structural information about the protein. Such structural information could be, e.g., the classic $\Phi\Psi$ angles [Schulz, 1979] that describe the relative orientation of peptide units along the protein backbone, or any other representation of local backbone structure. A classification of local structure into "secondary structure classes", is defined to be the result of any algorithm that uses a representation of local structure as Input, and which produces discrete classification labels as Output. This is a very general definition of local secondary structure that subsumes all previous definitions.

We develop classifications that are more predictable than the standard classifications [Pauling, 1951] [Kabsch, 1983] which were used in previous machine learning projects, as well as in other analyses of protein shape. We show that these new, predictable classes of secondary structure bear some relation to the conventional category of "helix", but also display significant differences.

We consider the definition, and prediction from sequence, of just two classes of structure. The extension to multiple classes is not difficult, but will not be made explicit here for reasons of clarity. We won't discuss details concerning construction of a representative training set, or details of conventional neural network training algorithms, such as backpropagation. These are well studied subjects that are addressed in e.g., [Stolorz, 1992] in the context of protein secondary structure prediction. We note in passing that one can employ complicated network architectures containing many output neurons (e.g. three output neurons for predicting alpha helix, beta chain, random coil), or many hidden units etc. (c.f. [Stolorz, 1992], [Qian, 1988], [Kneller, 1990]). However, explanatory figures presented in the next section employ only one output unit per net, and no hidden units, for clarity.

A widely adopted definition of protein secondary structure classes is due to Kabsch and Sander [Kabsch, 1983]. It has become conventional to use the Kabsch and Sander definition to define, via local structural information, three classes of secondary structure: alpha helix, beta strand, and a default class called random coil. The Kabsch and Sander alpha helix and beta strand classification captures in large part the classification first introduced by Pauling and Corey [Pauling, 1951]. Software implementing the Kabsch and Sander definitions, which take a local window of structural information as Input, and produce the Kabsch and Sander secondary structure classification of the window as Output, is widely available.

The key ideas of this paper are contained in Fig. (1).

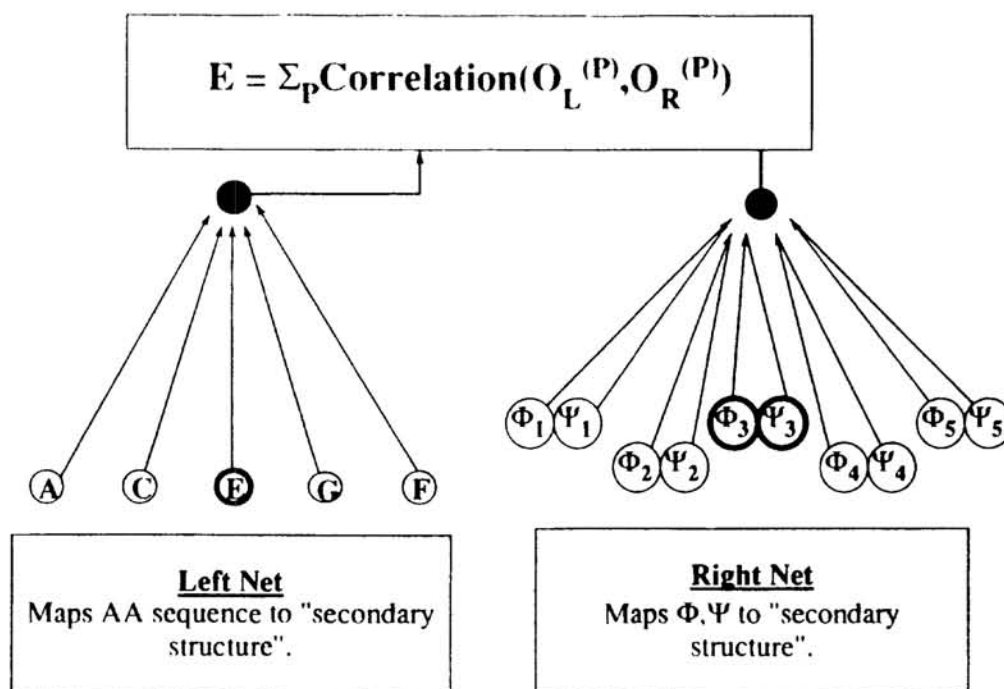

In this figure the Kabsch and Sander rules are represented by a second neural network. The Kabsch and Sander rules are just an Input/Output mapping (from a local window of structure to a classification of that structure) and may in principle be replaced with an equivalent neural net representing the same Input/Output mapping. We explicitly demonstrated that a simple neural net is capable of representing rules of the complexity of the Kabsch and Sander rules by training a network to perform the same structure classification as the Kabsch and Sander rules, and obtained high accuracy.

The representation of the structure data in the right-hand network uses $\Phi\Psi$ angles. The right-hand net sees a window of $\Phi\Psi$ angles corresponding to the window of amino acids in the left-hand network. Problems due to the angular periodicity of the $\Phi\Psi$ angles (i.e., 360 degrees and 0 degrees are different numbers, but represent the same angle) are eliminated by utilizing both the *sin* and *cos* of each angle.

The representation of the amino acids in the left–hand network is the usual unary representation employing twenty bits per amino acid. Results quoted in this paper do not use a special twenty-first bit to represent positions in a window extending past the ends of a protein.

Note that the right–hand neural network could implement extremely general definitions of secondary structure by changing the weights. We next show how to change the weights in a fashion so that new classifications of secondary structure are derived under the important restriction that they be predictable from amino acid sequence. In other words, we require that the synaptic weights be chosen so that the output of the left–hand network and the output of the right–hand network agree for each sequence–structure pair that is input to the two networks.

To achieve this, both networks are trained *simultaneously*, starting from random initial weights in each net, under the sole constraint that the outputs of the two networks agree for each pattern in the training set. The mathematical implementation of this constraint is described in various versions below. This procedure is a general, effective method of evolving *predictable* secondary structure classifications of experimental data. The goal of this research is to use two mutually self-supervised networks to define new classes of protein secondary structure which are more predictable from sequence than the standard classes of alpha helix, beta sheet or coil.

## 3   CONSTRAINING THE TWO NETS TO AGREE

One way to impose agreement between the outputs of the two networks is to require that they covary when viewed as a stream of real numbers. Note that it is not sufficient to merely require that the outputs of the left–hand and right–hand nets agree by, e.g., minimizing the following objective function

$$E = \sum_p (LeftO^{(p)} - RightO^{(p)})^2 \tag{1}$$

Here, $LeftO^{(p)}$ and $RightO^{(p)}$ represent the outputs of the left–hand and right–hand networks, respectively, for the $p^{th}$ pair of input windows: (sequence window –left net) and (structure window –right net). It is necessary to avoid the trivial minimum of $E$ obtained where the weights and thresholds are set so that each net presents a constant Output regardless of the input data. This is easily accomplished in Eqn (1) by merely setting all the weights and thresholds to 0.0.

Demanding that the outputs vary, or more explicitly co–vary, is a viable solution to avoiding trivial local minima. Therefore, one can maximize the correlation, $\rho$, between the left–hand and right–hand network outputs. The standard correlation measure between two objects, $LeftO^{(p)}$ and $RightO^{(p)}$ is:

$$\rho = \sum_p (LeftO^{(p)} - \overline{LeftO})(RightO^{(p)} - \overline{RightO}) \tag{3}$$

where $\overline{LeftO}$ denotes the mean of the left net's outputs over the training set, and respectively for the right net. $\rho$ is zero if there is no variation, and is maximized

if there is simultaneously both individual variation and joint agreement. In our situation it is equally desirable to have the networks maximally anti-correlated as it is for them to be correlated. (Whether the networks choose correlation, or anti-correlation, is evident from the behavior on the training set). Hence the minimization of $E = -\rho^2$ would ensure that the outputs are maximally correlated (or anti-correlated). While this work was in progress we received a preprint by Schmidhuber [Schmidhuber, 1992] who essentially implemented Eqn. (1) with an additional variance term (in a totally different context). Our results using this measure seem quite susceptible to local minima and we prefer alternative measures to enforce agreement.

One alternative to enforce agreement, since one ultimately measures predictive performance on the basis of the Mathews correlation coefficient (see, e.g., [Stolorz, 1992]), is to simultaneously train the two networks to maximize this measure. The Mathews coefficient, $C_i$, for the $i^{th}$ state is defined as:

$$C_i = \frac{p_i n_i - u_i o_i}{[(n_i + u_i)(n_i + o_i)(p_i + u_i)(p_i + o_i)]^{1/2}}$$

where $p_i$ is the number of examples where the left–hand net and right–hand net both predict class $i$, $n_i$ is the number of examples where neither net predicts $i$, $u_i$ counts the examples where the left net predicts $i$ and the right net does not, and $o_i$ counts the reverse. Minimizing $E = -C_i^2$ optimizes $C_i$.

Other training measures forcing agreement of the left and right networks may be used. Particularly suitable for the situation of many outputs (i.e., more than two–class discrimination) is "mutual information". Use of mutual information in this context is related to the IMAX algorithm for unsupervised detection of regularities across spatial or temporal data [Becker, 1992]. The mutual information is defined as

$$M = \sum_{i,j} p_{ij} \log \frac{p_{ij}}{p_{i.} p_{.j}} \tag{4}$$

where $p_{ij}$ is the joint probability of occurrence of the states of the left and right networks. (In previous work [Stolorz, 1992] we showed how $p_{ij}$ may be defined in terms of neural networks). Minimizing $E = -M$ maximizes $M$. While $M$ has many desirable properties as a measure of agreement between two or more variables [Stolorz, 1992] [Farber, 1992] [Lapedes, 1989] [Korber, 1993], our preliminary simulations show that maximizing $M$ is often prone to poor local maxima.

Finally, an alternative to using mutual information for multi-class, as opposed to dichotomous classification, is the Pearson correlation coefficient, $X^2$. This is defined in terms of $p_{ij}$ as

$$X^2 = \sum_{i,j} \frac{(p_{ij} - p_{i.} p_{.j})^2}{p_{i.} p_{.j}} \tag{5}$$

Our simulations indicate that $X^2$, $C_i$ and $\rho$ are all less susceptible to local minima

than $M$. However, these other objective functions suffer the defect that predictability is emphasized at the expense of utility. In other words, they can be maximal for the peculiar situation where a structural class is defined that occurs very rarely in the data, but when it occurs, it is predicted perfectly by the other network. The utility of this classification is therefore degraded by the fact that the predictable class only occurs rarely. Fortunately, this effect did not cause difficulties in the simulations we performed. Our best results to date have been obtained using the Mathews objective function (see Results).

## 4   RESULTS

The database we used consisted of 105 proteins and is identical to that used in previous investigations [Kneller, 1990] [Stolorz, 1992]. The proteins were divided into two groups: a set of 91 "training" proteins, and a distinct "prediction" set of 14 proteins. The resulting database is similar to the database used by Qian & Sejnowski [Qian, 1988] in their neural network studies of conventional secondary structure prediction. When comparison to predictability of conventional secondary structure classes was needed, we defined the conventional alpha, beta and coil states using the Kabsch and Sander definitions and therefore these states are identical to those used in previous work [Kneller, 1990] [Stolorz, 1992]. A window size of 13 residues resulted in 16028 train set examples and 3005 predict set examples. Effects of other windows sizes have not yet been extensively tested. All results, including conventional backpropagation training of Kabsch and Sander classifications, as well as two–net training of our new secondary structure classifications, did not employ an extra symbol denoting positions in a window that extended past the ends of a protein. Use of such a symbol could further increase accuracy.

We found that random initial conditions are necessary for the development of interesting new classes. However, random initial conditions also suffer to a certain extent from local minima. The mutual information function, in particular, often gets trapped quickly in uninteresting local minima when evolved from random initial conditions. More success was obtained with the other objective functions discussed above. We have not exhaustively investigated strategies to avoid local minima, and usually just chose new initial conditions if an uninteresting local minimum was encountered.

Results were best for two class discrimination using the Mathews objective function and a layer of five hidden units in each net. If one assigns the name "Xclass" to the newly defined structural class, then the Mathews coefficient on the prediction set for the Xclass dichotomy is $-0.425$. The Mathews coefficient on the train set for the Xclass dichotomy is $-0.508$. For comparison, the Mathews coefficient on the same predict set data for dichotomization (using standard backpropagation training with no hidden units), into the standard secondary structure classes Alpha/NotAlpha, Beta/NotBeta, and Coil/NotCoil is 0.33, 0.26, and 0.39, respectively. Adding hidden units gives negligible accuracy increase in predicting the conventional classes, but is important for improved prediction of the new classes. The negative sign of the two-net result indicates anti-correlation – a feature allowed by our objective function. The sign of the correlation is easily assessed on the train set and then can be trivially compensated for in prediction.

A natural question to ask is whether the new classes are simply related to the more conventional classes of alpha helix, beta, coil. A simple answer is to compute the Mathews correlation coefficient of the new secondary structure classes with each of the three Kabsch and Sander classes, for those examples in which the sequence network agreed with the structure network's classification. The correlation with Kabsch and Sander's alpha helix is highest: a Mathews coefficient of 0.248 was obtained on the train set, while a Mathews coefficient of 0.247 was obtained on the predict set. There is therefore a significant degree of correlation with the conventional classification of alpha helix, but significant differences exist as well. The new classes are a mixture of the conventional classes, and are not solely dominated by either alpha, beta or coil. Conventional alpha-helices comprise roughly 25% of the data (for both train and predict sets), while the new Xclass comprises 10%. It is quite interesting that an evolution of secondary structure classifications starting from random initial conditions, and hence completely unbiased towards the conventional classifications, results in a classification that has significant relationship to conventional helices but is more predictable from amino acid sequence than conventional helices. Graphical analysis (not shown here) of the new Xclass shows that the Xclass that is most closely related to helix typically extends the definition of helix past the standard boundaries of an alpha–helix.

## 5   CONCLUSIONS

A primary goal of this investigation is to evolve highly predictable secondary structure classes. Ultimately, such classes could be used, e.g., to provide constraints on tertiary structure calculations. Further work remains to derive even more predictable classes and to analyze their physical meaning. However, it is now clear that the use of two, co-evolving, adaptive networks defines a novel and useful machine learning paradigm that allows the evolution of new definitions of secondary structure that are significantly more predictable from primary amino acid sequence than the conventional definitions.

Related work is that of [Hunter, 1992], [Hunter, 1992], [Zhang, 1992], [Zhang, 1993] in which clustering either only in sequence space, or only in structure space, is attempted. However, no condition on the compatibility of the clustering is required, so new classes of structure are not guaranteed to be predictable from sequence.

Finally, we note that the methods described here might be usefully applied to other cognitive/perceptual or engineering tasks in which correlation of two or more different representations of the same data is required. In this regard the relation of our work to that of independent work of Becker [Becker, 1992], and of Schmidhuber [Schmidhuber, 1992], should be noted.

### Acknowledgements

We are grateful for useful discussions with Geoff Hinton, Sue Becker, and Joe Bryngelson. Sue Becker's contribution of software that was used in the early stages of this project is much appreciated. The research of Alan Lapedes and Robert Farber was supported in part by the U.S. Department of Energy. The authors would like to acknowledge the hospitality of the Santa Fe Institute, where much of this work

was performed.

# References

[Becker, 1992]     S. Becker. *An Information-theoretic Unsupervised Learning Algorithm for Neural Networks*. PhD thesis, University of Toronto (1992)

[Becker, 1992]     S. Becker, G. Hinton, *Nature* **355**, 161-163 (1992)

[Chou, 1978]       P. Chou, G. Fasman *Adv. Enzymol.* **47**, 45 (1978)

[Farber, 1992]     R. Farber, A. Lapedes *J. Mol. Biol.* **226** , 471, (1992)

[Hunter, 1992]     L. Hunter, N. Harris, D. States *Proceedings of the Ninth International Conference on Machine Learning*, San Mateo, California, Morgan Kaufmann Associates (1992)

[Hunter, 1992]     L. Hunter, D. States, *IEEE Expert*, **7**(4) 67-75 (1992)

[Kabsch, 1983]     W. Kabsch, C. Sander *Biopolymers* **22**, 2577 (1983)

[Kneller, 1990]    D. Kneller, F. Cohen, R. Langridge *J. Mol. Biol.* **214**, 171 (1990)

[Korber, 1993]     B. Korber, R. Farber, D. Wolpert, A. Lapedes *P.N.A.S. – in press* (1993)

[Lapedes, 1989]    A. Lapedes, C.Barnes, C. Burks, R.Farber, K. Sirotkin *in Computers and DNA* editors: G.Bell, T. Marr, (1989)

[Maclin, 1992]     R. Maclin, J. W. Shavlik *Proceedings of the Tenth National Conference on Artificial Intelligence*, San Jose, California, Morgan Kauffman Associates (1992)

[Pauling, 1951]    L. Pauling, R. Corey *Proc. Nat. Acad. Sci.* **37**, 205 (1951)

[Qian, 1988]       N. Qian, T. Sejnowski *J. Mol. Biol.* **202**, 865 (1988)

[Schmidhuber, 1992] J. Schmidhuber *Discovering Predictable Classifications*, Technical report CU-CS-626-92, Department of Computer Science, University of Colorado (1992)

[Schulz, 1979]     G. Schulz, R. Schirmer *Principles of Protein Structure* Springer Verlag, New York, (1979)

[Stolorz, 1992]    P. Stolorz, A. Lapedes, X. Yuan *J. Mol. Biol.* **225**, 363 (1992)

[Zhang, 1992]      X. Zhang, D. Waltz *in Artificial Intelligence and Molecular Biology*, editor: L. Hunter, AAAI Press (MIT Press) (1992)

[Zhang, 1993]      X. Zhang, J. Fetrow, W. Rennie, D. Waltz, G. Berg, in *Proceedings: First International Conference on Intelligent Systems For Molecular Biology*, p. 438, editors: L. Hunter, D. Searls, J. Shavlik, AAAI Press, Menlo Park, CA. (1993)
